# DIFFRAC : a discriminative and flexible framework for clustering

**Francis R. Bach**
INRIA - Willow Project
École Normale Supérieure
45, rue d'Ulm, 75230 Paris, France
`francis.bach@mines.org`

**Zaïd Harchaoui**
LTCI, TELECOM ParisTech and CNRS
46, rue Barrault
75634 Paris cedex 13, France
`zaid.harchaoui@enst.fr`

## Abstract

We present a novel linear clustering framework (DIFFRAC) which relies on a linear discriminative cost function and a convex relaxation of a combinatorial optimization problem. The large convex optimization problem is solved through a sequence of lower dimensional singular value decompositions. This framework has several attractive properties: (1) although apparently similar to K-means, it exhibits superior clustering performance than K-means, in particular in terms of robustness to noise. (2) It can be readily extended to non linear clustering if the discriminative cost function is based on positive definite kernels, and can then be seen as an alternative to spectral clustering. (3) Prior information on the partition is easily incorporated, leading to state-of-the-art performance for semi-supervised learning, for clustering or classification. We present empirical evaluations of our algorithms on synthetic and real medium-scale datasets.

## 1 Introduction

Many clustering frameworks have already been proposed, with numerous applications in machine learning, exploratory data analysis, computer vision and speech processing. However, these unsupervised learning techniques have not reached the level of sophistication of supervised learning techniques, that is, for all methods, there are still a significant number of explicit or implicit parameters to tune for successful clustering, most generally, the number of clusters and the metric or the similarity structure over the space of configurations.

In this paper, we present a **di**scriminative and **f**lexible **fra**mework for **c**lustering (DIFFRAC), which is aimed at alleviating some of those practical annoyances. Our framework is based on a recent set of works [1, 2] that have used the support vector machine (SVM) cost function used for linear classification as a clustering criterion, with the intuitive goal of looking for clusters which are most linearly separable. This line of work has led to promising results; however, the large convex optimization problems that have to be solved prevent application to datasets larger than few hundreds data points.[1] In this paper, we consider the maximum value of the regularized linear regression on indicator matrices. By choosing a square loss (instead of the hinge loss), we obtain a simple cost function which can be simply expressed in closed form and is amenable to specific efficient convex optimization algorithms, that can deal with large datasets of size 10,000 to 50,000 data points. Our cost function turns out to be a linear function of the "equivalence matrix" $M$, which is a square $\{0, 1\}$-matrix indexed by the data points, with value one for all pairs of data points that belong to the same clusters, and zero otherwise. In order to minimize this cost function with respect to $M$, we follow [1] and [2] by using convex outer approximations of the set of equivalence matrices, with a novel constraint on the minimum number of elements per cluster, which is based on the eigenvalues of $M$, and essential to the success of our approach.

In Section 2, we present a derivation of our cost function and of the convex relaxations. In Section 3, we show how the convex relaxed problem can be solved efficiently through a sequence of lower dimensional singular value decompositions, while in Section 4, we show how a priori knowledge can be incorporated into our framework. Finally, in Section 5, we present simulations comparing our new set of algorithms to other competing approaches.

## 2   Discriminative clustering framework

In this section, we first assume that we are given $n$ points $x_1, \ldots, x_n$ in $\mathbb{R}^d$, represented in a matrix $X \in \mathbb{R}^{n \times d}$. We represent the various partitions of $\{1, \ldots, n\}$ into $k > 1$ clusters by *indicator matrices* $y \in \{0, 1\}^{n \times k}$ such that $y1_k = 1_n$, where $1_k$ and $1_n$ denote the constant vectors of all ones, of dimensions $k$ and $n$. We let denote $\mathcal{I}_k$ the set of $k$-class indicator matrices.

### 2.1   Discriminative clustering cost

Given $y$, we consider the regularized linear regression problem of $y$ given $X$, which takes the form:

$$\min_{w \in \mathbb{R}^{d \times k},\, b \in \mathbb{R}^{1 \times k}} \frac{1}{n} \|y - Xw - 1_n b\|_F^2 + \kappa \operatorname{tr} w^\top w, \tag{1}$$

where the Frobenius norm is defined for any vector or rectangular matrix as $\|A\|_F^2 = \operatorname{tr} AA^\top = \operatorname{tr} A^\top A$. Denoting $f(x) = w^\top x + b \in \mathbb{R}^k$, this corresponds to a multi-label classification problem with square loss functions [4, 5]. The main advantage of this cost function is the possibility of (a) minimizing the regularized cost in closed form and (b) including a bias term by simply centering the data; namely, the global optimum is attained at $w^* = (X^\top \Pi_n X + n\kappa I_n)^{-1} X^\top \Pi_n y$ and $b^* = \frac{1}{n} 1_n^\top (y - Xw^*)$, where $\Pi_n = I_n - \frac{1}{n} 1_n 1_n^\top$ is the usual centering projection matrix. The optimal value is then equal to

$$J(y, X, \kappa) = \operatorname{tr} yy^\top A(X, \kappa), \tag{2}$$

where the $n \times n$ matrix $A(X, \kappa)$ is defined as:

$$A(X, \kappa) = \frac{1}{n} \Pi_n (I_n - X(X^\top \Pi_n X + n\kappa I)^{-1} X^\top) \Pi_n. \tag{3}$$

The matrix $A(X, \kappa)$ is positive semi-definite, i.e., for all $u \in \mathbb{R}^n$, $u^\top A(X, \kappa)u \geqslant 0$, and $1_n$ is a singular vector of $A(X, \kappa)$, i.e., $A(X, \kappa)1_n = 0$.

Following [1] and [2], we are thus looking for a $k$-class indicator matrix $y$ such that $\operatorname{tr} yy^\top A(X, \kappa)$ is minimal, i.e., for a partition such that the clusters are most linearly separated, where the separability of clusters is measured through the minimum of the discriminative cost with respect to all linear classifiers. This combinatorial optimization is NP-hard in general [6], but efficient convex relaxations may be obtained, as presented in the next section.

### 2.2   Indicator and equivalence matrices

The cost function defined in Eq. (2) only involves the matrix $M = yy^\top \in \mathbb{R}^{n \times n}$. We let denote $\mathcal{E}_k$ the set of "$k$-class equivalence matrices", i.e., the set of matrices $M$ such that there exists a $k$-class indicator matrix $y$ with $M = yy^\top$.

There are many outer convex approximations of the discrete sets $\mathcal{E}_k$, based on different properties of matrices in $\mathcal{E}_k$, that were used in different contexts, such as maximum cut problems [6] or correlation clustering [7]. We have the following usual properties of equivalence matrices (independent of $k$): if $M \in \mathcal{E}_k$, then (a) $M$ is positive semidefinite (denoted as $M \succcurlyeq 0$), (b) $M$ has nonnegative values (denoted as $M \geqslant 0$), and (c) the diagonal of $M$ is equal to $1_n$ (denoted as $\operatorname{diag}(M) = 1_n$).

Moreover, if $M$ corresponds to at most $k$ clusters, we have $M \succcurlyeq \frac{1}{k} 1_n 1_n^\top$, which is a consequence to the convex outer approximation of [6] for the maximum $k$-cut problem. We thus use the following convex outer approximation:

$$\mathcal{C}_k = \{M \in \mathbb{R}^{n \times n}, \, M = M^\top, \, \operatorname{diag}(M) = 1_n, \, M \geqslant 0, \, M \succcurlyeq \tfrac{1}{k} 1_n 1_n^\top\} \supset \mathcal{E}_k.$$

Note that when $k = 2$, the constraints $M \geqslant 0$ (pointwise nonnegativity) is implied by the other constraints.

## 2.3 Minimum cluster sizes

Given the discriminative nature of our cost function (and in particular that $A(X, \kappa)1_n = 0$), the minimum value 0 is always obtained with $M = 1_n 1_n^\top$, a matrix of rank one, equivalent to a single cluster. Given the number of desired clusters, we thus need to add some prior knowledge regarding the size of those clusters. Following [1], we impose a minimum size $\lambda_0$ for each cluster, through row sums and eigenvalues:

**Row sums** If $M \in \mathcal{E}_k$, then $M1_n \geqslant \lambda_0 1_n$ and $M1_n \leqslant (n - (k-1)\lambda_0)1_n$ (the cluster must be smaller than $(n - (k-1)\lambda_0)$ if they are all larger than $\lambda_0$)–this is the same constraint as in [1].

**Eigenvalues** When $M \in \mathcal{E}_k$, the sizes of the clusters are exactly the $k$ largest eigenvalues of $M$. Thus, for a matrix in $\mathcal{E}_k$, the minimum cluster size constraint is equivalent to $\sum_{i=1}^n 1_{\lambda_i(M) \geqslant \lambda_0} \geqslant k$, where $\lambda_1(M), \ldots, \lambda_n(M)$ are the $n$ eigenvalues of $M$. Functions of the form $\Phi(M) = \sum_{i=1}^n \phi(\lambda_i(M))$ are referred to as *spectral functions* and are particularly interesting in machine learning and optimization, since $\Phi$ inherits from $\phi$ many of its properties, such as differentiability and convexity [8]. The previous constraint can be seen as $\Phi(M) \geqslant k$, with $\phi(\lambda) = 1_{\lambda \geqslant \lambda_0}$, which is not concave and thus does not lead to a convex constraint. In this paper we propose to use the concave upper envelope of this function, namely $\phi_{\lambda_0}(\lambda) = \min\{\lambda/\lambda_0, 1\}$, thus leading to a novel additional constraint.

Our final convex relaxation is thus of minimizing $\mathrm{tr} A(X, \kappa)M$ with respect to $M \in \mathcal{C}_k$ and such that $\Phi_{\lambda_0}(M) \geqslant k$, $M1_n \geqslant \lambda_0 1_n$ and $M1_n \leqslant (n - (k-1)\lambda_0)1_n$, where $\Phi_{\lambda_0}(M) = \sum_{i=1}^n \min\{\lambda_i(M)/\lambda_0, 1\}$. The clustering results are empirically robust to the value of $\lambda_0$. In all our simulations we use $\lambda_0 = \lfloor n/2k \rfloor$.

## 2.4 Comparison with K-means

Our method bears some resemblance with the usual $K$-means algorithm. Indeed, in the unregularized case ($\kappa = 0$), we aim to minimize

$$\mathrm{tr}\, \Pi_n(I_n - X(X^\top \Pi_n X)^{-1} X^\top)\Pi_n yy^\top.$$

Results from [9] show that K-means aims at minimizing the following criterion with respect to $y$:

$$\min_{\mu \in \mathbb{R}^{k \times d}} \|X - y\mu\|_F^2 = \mathrm{tr}(I_n - y(y^\top y)^{-1} y^\top)(\Pi_n X)(\Pi_n X)^\top.$$

The main differences between the two cost functions are that (1) we require an additional parameter, namely the minimum number of elements per cluster and (2) our cost function normalizes the data, while the K-means distortion measure normalizes the labels. This apparently little difference has a significant impact on the performance, as our method is invariant by affine scaling of the data, while K-means is only invariant by translation, isometries and isotropic scaling, and is very much dependent on how the data are presented (in particular the marginal scaling of the variables). In Figure 1, we compare the two algorithms on a simple synthetic task with noisy dimensions, showing that ours is more robust to noisy features. Note that using a discriminative criterion based on the square loss may lead to the *masking problem* [4], which can be dealt with in the usual way by using second-order polynomials or, equivalently, a polynomial kernel.

## 2.5 Kernels

The matrix $A(X, \kappa)$ in Eq. (3) can be expressed only in terms of the Gram matrix $K = XX^\top$. Indeed, using the matrix inversion lemma, we get:

$$A(K, \kappa) = \kappa \Pi_n (\widetilde{K} + n\kappa I_n)^{-1} \Pi_n, \tag{4}$$

where $\widetilde{K} = \Pi_n K \Pi_n$ is the "centered Gram matrix" of the points $X$. We can thus apply our framework with any positive definite kernel [5].

## 2.6 Additional relaxations

Our convex optimization problem can be further relaxed. An interesting relaxation is obtained by (1) relaxing the constraints $M \succcurlyeq \frac{1}{k} 1_n 1_n^\top$ into $M \succcurlyeq 0$, (2) relaxing $\mathrm{diag}(M) = 1_n$ into $\mathrm{tr} M = n$,

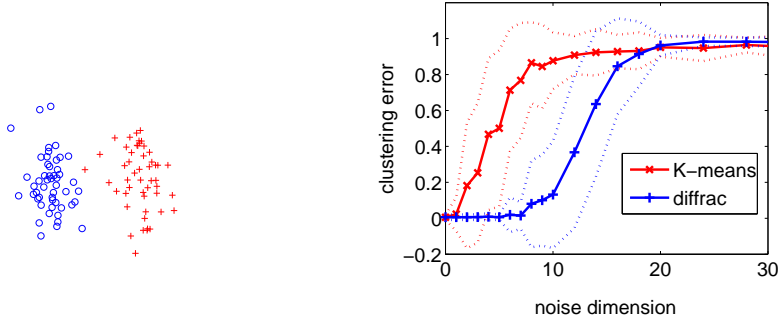

Figure 1: Comparison with K-means, on a two-dimensional dataset composed of two linearly separable bumps (100 data points, plotted in the left panel), with additional random independent noise dimensions (with normal distributions with same marginal variances as the 2D data). The clustering performance is plotted against the number of irrelevant dimensions, for regular K-means and our DIFFRAC approach (right panel, averaged over 50 replications with the standard deviation in dotted lines) . The clustering performance is measured by a metric between partitions defined in Section 5, which is always between 0 and 1.

and (3) removing the constraint $M \geqslant 0$ and the constraints on the row sums. A short calculation shows that this relaxation leads to an eigenvalue problem: let $A = \sum_{i=1}^{n} a_i u_i u_i^{\top}$ be an eigenvalue decomposition of $A$, where $a_1 \leqslant \cdots \leqslant a_n$ are the sorted eigenvalues. The minimal value of the relaxed convex optimization problem is attained at $M = \sum_{i=1}^{j} u_i u_i^{\top} + (n - \lambda_0 j) u_{j+1} u_{j+1}^{\top}$, with $j = \lfloor n/\lambda_0 \rfloor$. This additional relaxation into an eigenvalue problem is the basis of our efficient optimization algorithm in Section 3.

In the kernel formulation, since the smallest eigenvectors of $A = \frac{1}{n} \Pi_n (\widetilde{K} + n\kappa I_n)^{-1} \Pi_n$ are the same as the largest eigenvectors of $\widetilde{K}$, the relaxed problem is thus equivalent to kernel principal component analysis [10, 5] in the kernel setting, and in the linear setting to regular PCA (followed by our rounding procedure presented in Section 3.3). In the linear setting, since PCA has no clustering effects in general[2], it is clear that the constraints that were removed are essential to the clustering performance. In the kernel setting, experiments have shown that the most important constraint to keep in order to achieve the best embedding and clustering is the constraint $\mathrm{diag}(M) = 1_n$.

## 3  Optimization

Since $\phi_{\lambda_0}(\lambda) = \frac{1}{2\lambda_0}(\lambda + \lambda_0 - |\lambda - \lambda_0|)$, and the sum of singular values can be represented as a semidefinite program (SDP), our problem is an SDP. It can thus be solved to any given accuracy in polynomial time by general purpose interior-point methods [12]. However, the number of variables is $O(n^2)$ and thus the complexity of general purpose algorithms will be at least $O(n^7)$; this remains much too slow for medium scale problems, where the number of data points is between 1,000 and 10,000. We now present an efficient approximate method that uses the specificity of the problem to reduce the computational load.

### 3.1  Optimization by partial dualization

We saw earlier that by relaxing some of the constraints, we get back an eigenvalue problem. Eigenvalue decompositions are among the most important tools in numerical algebra and algorithms and codes are heavily optimized for these, and it is thus advantageous to rely on a sequence of eigenvalue decompositions for large scale algorithms.

We can dualize some constraints while keeping others; this leads to the following proposition:

**Proposition 1** *The solution of the convex optimization problem defined in Section 2.3 can be obtained my maximizing* $F(\beta) = \min_{M \succcurlyeq 0, \mathrm{tr} M = n, \Phi_{\lambda_0}(M) \geqslant k} \mathrm{tr} B(\beta) M - b(\beta)$ *with respect to* $\beta$*, where*

$$B(\beta) = A + \mathrm{Diag}(\beta_1) - \frac{1}{2}(\beta_2 - \beta_3)1^\top - \frac{1}{2}1(\beta_2 - \beta_3)^\top - \beta_4 + \frac{1}{2}\frac{\beta_5 \beta_5^\top}{\beta_6}$$

$$b(\beta) = \beta_1^\top 1 - (n - (k-1)\lambda_0)\beta_2^\top 1 + \lambda_0 \beta_3^\top 1 + k\beta_6/2 + \beta_5^\top 1,$$

*and* $\beta_1 \in \mathbb{R}^n$, $\beta_2 \in \mathbb{R}_+^n$, $\beta_3 \in \mathbb{R}_+^n$, $\beta_4 \in \mathbb{R}_+^{n \times n}$, $\beta_5 \in \mathbb{R}^n$, $\beta_6 \in \mathbb{R}_+$.

The variables $\beta_1$, $\beta_2$, $\beta_3$, $\beta_4$, $(\beta_5, \beta_6)$ correspond to the respective dualizations of the constraints $\mathrm{diag}(M) = 1_n$, $M1_n \leqslant (n - (k-1)\lambda_0)1_n$, $M1_n \geqslant \lambda_0 1_n$, $M \geqslant 0$, and $M \succcurlyeq \frac{1_n 1_n^\top}{k}$.

The function $J(B) = \min_{M \succcurlyeq 0, \mathrm{tr} M = n, \Phi_{\lambda_0}(M) \geqslant k} \mathrm{tr} BM$ is a spectral convex function and may be computed in closed form through an eigenvalue decomposition. Moreover, a subgradient may be easily computed, readily leading to a numerically efficient subgradient method in fewer dimensions than $n^2$. Indeed, if we subsample the pointwise positivity constraint $N \geqslant 0$ (so that $\beta_4$ has only a size smaller than $n^{1/2} \times n^{1/2}$), then the set of dual variables $\beta$ we are trying to maximize has linear size in $n$ (instead of the primal variable $M$ being quadratic in $n$).

More refined optimization schemes, based on smoothing of the spectral function $J(B)$ by $\min_{M \succcurlyeq 0, \mathrm{tr} M = n, \Phi_{\lambda_0}(M) \geqslant k}[\mathrm{tr} BM + \varepsilon \mathrm{tr} M^2]$ are also used to speed up convergence (steepest descent of a smoothed function is generally faster than subgradient iterations) [13].

### 3.2 Computational complexity

The running time complexity can be splitted into initialization procedures and per iteration complexity. The per iteration complexity depends directly on the cost of our eigenvalue problems, which themselves are linear in the matrix-vector operation with the matrix $A$ (we only require a fixed small number of eigenvalues). In all situations, we manage to keep a linear complexity in the number $n$ of data points. Note, however, that the number of descent iterations cannot be bounded a priori; in simulations we limit the number of those iterations to 200.

For linear kernels with dimension $d$, the complexity of initialization is $O(d^2 n)$, while the complexity of each iteration is proportional to the cost of performing a matrix-vector operation with $A$, that is, $O(dn)$. For general kernels, the complexity of initialization is $O(n^3)$, while the complexity of each iteration is $O(n^2)$. However, using an incomplete Cholesky decomposition [5] makes all costs linear in $n$.

### 3.3 Rounding

After the convex optimization, we obtain a low-rank matrix $M \in \mathcal{C}_k$ which is pointwise nonnegative with unit diagonal, of the form $UU^\top$ where $U \in \mathbb{R}^{n \times m}$. We need to project it back to the discrete $\mathcal{E}_k$. We have explored several possibilities, all with similar results. We propose the following procedure: we first project $M$ back to the set of matrices of rank $k$ and unit diagonal, by computing an eigendecomposition, rescaling the first $k$ eigenvectors to unit norms and then perform K-means, which is equivalent to performing the spectral clustering algorithm of [14] on the matrix $M$.

## 4 Semi-supervised learning

Working with equivalence matrices $M$ allows to easily include prior knowledge on the clusters [2, 15, 16], namely, "must-link" constraints (also referred to a positive constraints) for which we constrain an element of $M$ to be one, and "must-not-link" constraints (also referred to as negative constraints), for which we constrain an element of $M$ to be zero. Those two constraints are linear in $M$ and can thus easily be included in our convex formulation.

We assume throughout this section that we have a set of "must-link" pairs $\mathcal{P}_+$ and a set of "must-not-link" pairs $\mathcal{P}_-$. Moreover, we assume that the set of positive constraints is *closed*, i.e., that if there is a path of positive constraints between two data points, then these two data points are already forming a pair in $\mathcal{P}_+$. If the set of positive pairs does not satisfy this assumption, a larger set of pairs can be obtained by transitive closure.

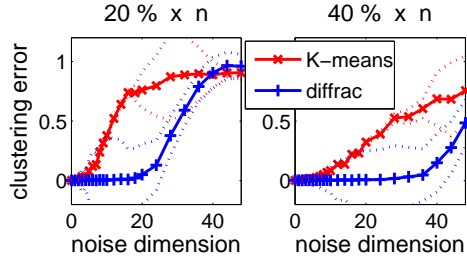

Figure 2: Comparison with K-means in the semi-supervised setting, with data taken from Figure 1: clustering performance (averaged over 50 replications, with standard deviations in dotted) vs. number of irrelevant dimensions, with $20\% \times n$ and $40\% \times n$ random matching pairs used for semi-supervision.

**Positive constraints**   Given our closure assumption on $\mathcal{P}_+$, we get a partition of $\{1,\ldots,n\}$ into $p$ "chunks" of size greater or equal to 1. The singletons in this partition correspond to data points that are not involved in any positive constraints, while other subsets corresponds to chunks of data points that must occur together in the final partition. We let $C_j$, $j = 1,\ldots,p$ denote those groups, and let $P$ denote the $n \times p$ $\{0,1\}$-matrix defined such that each column (indexed by $j$) is equal to one for rows in $C_j$ and zero otherwise. Forcing those groups is equivalent to considering $M$ of the form $M = P M_P P^\top$, where $M_P$ is an equivalence matrix of size $p$. Note that the positive constraint $M_{ij} = 1$ is in fact turned into the equality of columns (and thus rows by symmetry) $i$ and $j$ of $M$, which is equivalent when $M \in \mathcal{E}_k$, but much stronger for $M \in \mathcal{C}_k$.

In our linear clustering framework, this is in fact equivalent to (a) replacing each chunk by its mean, (b) adding a weight equal to the number of elements in the group into the discriminative cost function and (c) modifying the regularization matrix to take into account the inner variance within each chunk. Positive constraints can be similarly included into K-means, to form a reduced weighted K-means problem, which is simpler than other approaches to deal with positive constraints [17].

In Figure 2, we compare constrained K-means and the DIFFRAC framework under the same setting as in Figure 1, with different numbers of randomly selected positive constraints.

**Negative constraints**   After the chunks corresponding to positive constraints have been collapsed to one point, we extend the set of negative constraints to those collapsed points (if the constraints were originally consistent, the negative constraints can be uniquely extended). In our optimization framework, we simply add a penalty function of the form $\frac{1}{\varepsilon|\mathcal{P}_-|}\sum_{(i,j)\in\mathcal{P}_-} M_{ij}^2$. The K-means rounding procedure also has to be constrained, e.g., using the procedure of [17].

## 5   Simulations

In this section, we apply the DIFFRAC framework to various clustering problems and situations. In all our simulations, we use the following distance between partitions $B = B_1 \cup \cdots \cup B_k$ and $B' = B'_1 \cup \cdots \cup B'_{k'}$ into $k$ and $k'$ disjoints subsets of $\{1,\ldots,n\}$: $d(B,B') = \left(k + k' - 2\sum_{i,i'} \frac{\mathrm{Card}(B_i \cap B'_{i'})^2}{\mathrm{Card}(B_i)\mathrm{Card}(B'_{i'})}\right)^{1/2}$. $d(B,B')$ defines a distance over the set of partitions [9] which is always between 0 and $(k + k' - 2)^{1/2}$. When comparing partitions, we use the squared distance $\frac{1}{2}d(B,B')^2$, which is always between 0 and $\frac{k+k'}{2} - 1$ (and between 0 and $k - 1$, if the two partitions have the same number of clusters).

### 5.1   Clustering classification datasets

We looked at the Isolet dataset (26 classes, 5,200 data points) from the UCI repository and the MNIST datasets of handwritten digits (10 classes, 5,000 data points). For each of those datasets, we compare the performances of K-means, RCA [18] and DIFFRAC, for linear and Gaussian kernels (referred to as "rbf"), for fixed value of the regularization parameter, with different levels of supervision. Results are presented in Table 1: on unsupervised problems, K-means and DIFFRAC

| Dataset | K-means | DIFFRAC | RCA |
|---|---|---|---|
| Mnist-linear 0% | $\mathbf{5.6 \pm 0.1}$ | $6.0 \pm 0.4$ | |
| Mnist-linear 20% | $4.5 \pm 0.3$ | $3.6 \pm 0.3$ | $\mathbf{3.0 \pm 0.2}$ |
| Mnist-linear 40% | $2.9 \pm 0.3$ | $2.2 \pm 0.2$ | $\mathbf{1.8 \pm 0.4}$ |
| Mnist-RBF 0% | $5.6 \pm 0.2$ | $\mathbf{4.9 \pm 0.2}$ | |
| Mnist-RBF 20% | $4.6 \pm 0.0$ | $\mathbf{1.8 \pm 0.4}$ | $4.1 \pm 0.2$ |
| Mnist-RBF 40% | $4.9 \pm 0.0$ | $\mathbf{0.9 \pm 0.1}$ | $2.9 \pm 0.1$ |
| Isolet-linear 0% | $\mathbf{12.1 \pm 0.6}$ | $12.3 \pm 0.3$ | |
| Isolet-linear 20% | $10.5 \pm 0.2$ | $\mathbf{7.8 \pm 0.8}$ | $9.5 \pm 0.4$ |
| Isolet-linear 40% | $9.2 \pm 0.5$ | $\mathbf{3.7 \pm 0.2}$ | $7.0 \pm 0.4$ |
| Isolet-RBF 0% | $11.4 \pm 0.4$ | $\mathbf{11.0 \pm 0.3}$ | |
| Isolet-RBF 20% | $10.6 \pm 0.0$ | $\mathbf{7.5 \pm 0.5}$ | $7.8 \pm 0.5$ |
| Isolet-RBF 40% | $10.0 \pm 0.0$ | $\mathbf{3.7 \pm 1.0}$ | $6.9 \pm 0.6$ |

Table 1: Comparison of K-means, RCA and linear DIFFRAC, using the clustering metric defined in Section 5 (averaged over 10 replications), for linear and "rbf" kernels and various levels of supervision.

have similar performance, while on semi-supervised problems, and in particular for nonlinear kernels, DIFFRAC outperforms both K-means and RCA. Note that all algorithms work on the same data representation (linear or kernelized) and that differences are due to the underlying clustering frameworks.

## 5.2 Semi-supervised classification

To demonstrate the effectiveness of our method in a semi-supervised learning (SSL) context, we performed experiments on some benchmarks datasets for SSL described in [19]. We considered the following datasets: COIL, BCI and Text. We carried out the experiments in a transductive setting, i.e., the test set coincides with the set of unlabelled samples. This allowed us to conduct a fair comparison with the low density separation (LDS) algorithm of [19], which is an enhanced version of the so-called Transductive SVM. However, deriving "out-of-sample" extensions for our method is straightforward.

A primary goal in semi-supervised learning is to take into account a large number of labelled points in order to dramatically reduce the number of labelled points required to achieve a competitive classification accuracy. Henceforth, our experimental setting consists in observing how fast the classification accuracy collapses as the number of labelled points increases. The less labelled points a method needs to achieve decent classification accuracy, the more it is relevant for semi-supervised learning tasks. As shown in Figure 3, our method yields competitive classification accuracy with very few labelled points on the three datasets. Moreover, DIFFRAC reaches unexpectedly good results on the Text dataset, where most semi-supervised learning methods usually show disappointing performance. One explanation might be that DIFFRAC acts as an "augmented"-clustering algorithm, whereas most semi-supervised learning algorithms are built as "augmented"-versions of traditional supervised learning algorithms such as LDS which is built on SVMs for instance. Hence, for datasets exhibiting multi-class structure such as Text, DIFFRAC is more able to utilize unlabelled points since it based on a multi-class clustering algorithm rather than algorithms based on binary SVMs, where multi-class extensions are currently unclear. Thus, our experiments support the fact that semi-supervised learning algorithms built on clustering algorithms augmented with labelled data acting as hints on clusters are worth for investigation and further research.

## 6 Conclusion

We have presented a discriminative framework for clustering based on the square loss and penalization through spectral functions of equivalence matrices. Our formulation enables the easy incorporation of semi-supervised constraints, which leads to state-of-the-art performance in semi-supervised learning. Moreover, our discriminative framework should allow to use existing methods for learning the kernel matrix from data [20]. Finally, we are currently investigating the use of DIFFRAC in semi-supervised image segmentation. In particular, early experiments on estimating the number of clusters using variation rates of our discriminative costs are very promising.

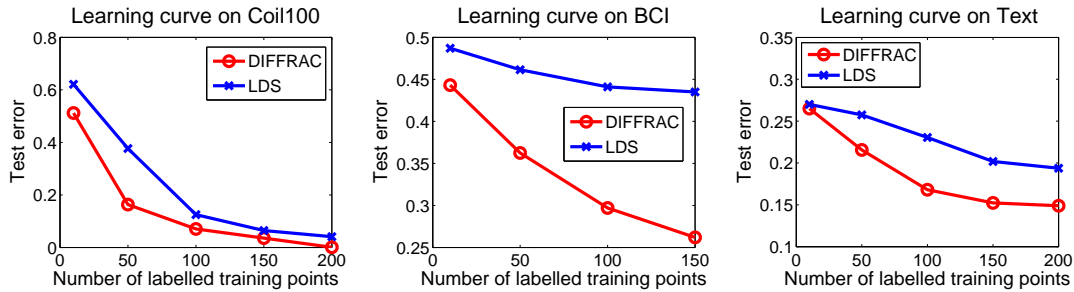

Figure 3: Semi-supervised classification.

## Footnotes

[1]Recent work [3] has looked at more efficient formulations.

[2]Recent results show however that it does have an effect when clusters are spherical Gaussians [11].

## References

[1] L. Xu, J. Neufeld, B. Larson, and D. Schuurmans. Maximum margin clustering. In *Adv. NIPS*, 2004.

[2] T. De Bie and N. Cristianini. Fast SDP relaxations of graph cut clustering, transduction, and other combinatorial problems. *J. Mac. Learn. Res.*, 7:1409–1436, 2006.

[3] K. Zhang, I. W. Tsang, and J. T. Kwok. Maximum margin clustering made practical. In *Proc. ICML*, 2007.

[4] T. Hastie, R. Tibshirani, and J. Friedman. *The Elements of Statistical Learning*. Springer-Verlag, 2001.

[5] J. Shawe-Taylor and N. Cristianini. *Kernel Methods for Pattern Analysis*. Camb. Univ. Press, 2004.

[6] A. Frieze and M. Jerrum. Improved approximation algorithms for MAX k-CUT and MAX BISECTION. In *Integer Programming and Combinatorial Optimization*, volume 920, pages 1–13. Springer, 1995.

[7] C. Swamy. Correlation clustering: maximizing agreements via semidefinite programming. In *ACM-SIAM Symp. Discrete algorithms*, 2004.

[8] A. S. Lewis and H. S. Sendov. Twice differentiable spectral functions. *SIAM J. Mat. Anal. App.*, 23(2):368–386, 2002.

[9] F R. Bach and M I. Jordan. Learning spectral clustering, with application to speech separation. *J. Mac. Learn. Res.*, 7:1963–2001, 2006.

[10] B. Schölkopf, A. J. Smola, and K.-R. Müller. Nonlinear component analysis as a kernel eigenvalue problem. *Neural Comp.*, 10(3):1299–1319, 1998.

[11] N. Srebro, G. Shakhnarovich, and S. Roweis. An investigation of computational and informational limits in gaussian mixture clustering. In *Proc. ICML*, 2006.

[12] S. Boyd and L. Vandenberghe. *Convex Optimization*. Camb. Univ. Press, 2003.

[13] J. F. Bonnans, J. C. Gilbert, C. Lemaréchal, and C. A. Sagastizbal. *Numerical Optimization Theoretical and Practical Aspects*. Springer, 2003.

[14] A. Y. Ng, M. I. Jordan, and Y. Weiss. On spectral clustering: analysis and an algorithm. In *Adv. NIPS*, 2002.

[15] L. Xu and D. Schuurmans. Unsupervised and semi-supervised multi-class support vector machines. In *Proc. AAAI*, 2005.

[16] M. Heiler, J. Keuchel, and C. Schnörr. Semidefinite clustering for image segmentation with a-priori knowledge. In *Pattern Recognition, Proc. DAGM*, 2005.

[17] K. Wagstaff, C. Cardie, S. Rogers, and S. Schrödl. Constrained K-means clustering with background knowledge. In *Proc. ICML*, 2001.

[18] A. Bar-Hillel, T. Hertz, N. Shental, and D. Weinshall. Learning distance functions using equivalence relations. In *Proc. ICML*, 2003.

[19] O. Chapelle and A. Zien. Semi-supervised classification by low density separation. In *Proc. AISTATS*, 2004.

[20] F. R. Bach, G. R. G. Lanckriet, and M. I. Jordan. Multiple kernel learning, conic duality, and the SMO algorithm. In *Proc. ICML*, 2004.
